# Construction of Dependent Dirichlet Processes based on Poisson Processes

**Dahua Lin**
CSAIL, MIT
dhlin@mit.edu

**Eric Grimson**
CSAIL, MIT
welg@csail.mit.edu

**John Fisher**
CSAIL, MIT
fisher@csail.mit.edu

## Abstract

We present a novel method for constructing dependent Dirichlet processes. The approach exploits the intrinsic relationship between Dirichlet and Poisson processes in order to create a Markov chain of Dirichlet processes suitable for use as a prior over evolving mixture models. The method allows for the creation, removal, and location variation of component models over time while maintaining the property that the random measures are marginally DP distributed. Additionally, we derive a Gibbs sampling algorithm for model inference and test it on both synthetic and real data. Empirical results demonstrate that the approach is effective in estimating dynamically varying mixture models.

## 1 Introduction

As the cornerstone of Bayesian nonparametric modeling, Dirichlet processes (DP) [22] have been applied to a wide variety of inference and estimation problems [3, 10, 20] with Dirichlet process mixtures (DPMs) [15, 17] being one of the most successful. DPMs are a generalization of finite mixture models that allow an indefinite number of mixture components. The traditional DPM model assumes that each sample is generated independently from the same DP. This assumption is limiting in cases when samples come from many, yet dependent, DPs. HDPs [23] partially address this modeling aspect by providing a way to construct multiple DPs implicitly depending on each other via a common parent. However, their hierarchical structure may not be appropriate in some problems (e.g. temporally varying DPs).

Consider a document model where each document is generated under a particular topic and each topic is characterized by a distribution over words. Over time, topics change: some old topics fade while new ones emerge. For each particular topic, the word distribution may evolve as well. A natural approach to model such topics is to use a Markov chain of DPs as a prior, such that the DP at each time is generated by varying the previous one in three possible ways: *creating a new topic*, *removing an existing topic*, and *changing the word distribution of a topic*.

Since MacEachern introduced the notion of dependent Dirichlet processes (DDP) [12], a variety of DDP constructions have been developed, which are based on either weighted mixtures of DPs [6, 14, 18], generalized Chinese restaurant processes [4, 21, 24], or the stick breaking construction [5, 7]. Here, we propose a fundamentally different approach, taking advantage of the intrinsic relationship between Dirichlet processes and Poisson processes: *a Dirichlet process is a normalized Gamma process, while a Gamma process is essentially a compound Poisson process.* The key idea is motivated by the following: observations that preserve complete randomness when applied to Poisson processes result in a new process that remains Poisson. Consequently, one can obtain a Dirichlet process which is dependent on other DPs by applying such operations to their underlying compound Poisson processes. In particular, we discuss three specific operations: *superposition*, *subsampling*, and *point transition*. We develop a Markov chain of DPs by combining these operations, leading to a framework that allows creation, removal, and location variation of particles. This

construction inherently comes with an elegant property that the random measure at each time is marginally DP distributed. Our approach relates to previous efforts in constructing dependent DPs while overcoming inherent limitations. A detailed comparison is given in section 4.

## 2   Poisson, Gamma, and Dirichlet Processes

Our construction of *dependent Dirichlet processes* rests upon the connection between Poisson, Gamma, and Dirichlet processes, as well as the concept of *complete randomness*. We briefly review these concepts; Kingman [9] provides a detailed exposition of the relevant theory.

Let $(\Omega, \mathcal{F}_\Omega)$ be a measurable space, and $\Pi$ be a random point process on $\Omega$. Each realization of $\Pi$ uniquely corresponds to a counting measure $N_\Pi$ defined by $N_\Pi(A) \triangleq \#(\Pi \cap A)$ for each $A \in \mathcal{F}_\Omega$. Hence, $N_\Pi$ is a measure-valued random variable or simply a *random measure*. A *Poisson process* $\Pi$ on $\Omega$ with mean measure $\mu$, denoted $\Pi \sim \mathrm{PoissonP}(\mu)$, is defined to be a point process such that $N_\Pi(A)$ has a Poisson distribution with mean $\mu(A)$ and that for any disjoint measurable sets $A_1, \ldots, A_n$, $N_\Pi(A_1), \ldots, N_\Pi(A_n)$ are independent. The latter property is referred to as *complete randomness*. Poisson processes are the only point process that satisfies this property [9]:

**Theorem 1.** *A random point process $\Pi$ on a regular measure space is a Poisson process if and only if $N_\Pi$ is completely random. If this is true, the mean measure is given by $\mu(A) = \mathbb{E}(N_\Pi(A))$.*

Consider $\Pi^* \sim \mathrm{PoissonP}(\mu^*)$ on a product space $\Omega \times \mathbb{R}^+$. For each realization of $\Pi^*$, We define $\Sigma^* : \mathcal{F}_\Omega \to [0, +\infty]$ as

$$\Sigma^* \triangleq \sum_{(\theta, w_\theta) \in \Pi^*} w_\theta \delta_\theta \tag{1}$$

Intuitively, $\Sigma^*(A)$ sums up the values of $w_\theta$ with $\theta \in A$. Note that $\Sigma^*$ is also a completely random measure (but not a point process in general), and is essentially a generalization of the *compound Poisson process*. As a special case, if we choose $\mu^*$ to be

$$\mu^* = \mu \times \gamma \quad \text{with } \gamma(dw) = w^{-1}e^{-w}dw, \tag{2}$$

Then the random measure as defined in Eq.(1) is called a *Gamma process* with base measure $\mu$, denoted by $G \sim \Gamma\mathrm{P}(\mu)$. Normalizing any realization of $G \sim \Gamma\mathrm{P}(\mu)$ yields a sample of a *Dirichlet process*, as

$$D \triangleq G/G(\Omega) \sim \mathrm{DP}(\mu). \tag{3}$$

In conventional parameterization, $\mu$ is often decomposed into two parts: a base distribution $p_\mu \triangleq \mu/\mu(\Omega)$, and a concentration parameter $\alpha_\mu \triangleq \mu(\Omega)$.

## 3   Construction of Dependent Dirichlet Processes

Motivated by the relationship between Poisson and Dirichlet processes, we develop a new approach for constructing dependent Dirichlet processes (DDPs). Our approach can be described as follows: given a collection of Dirichlet processes, one can apply operations that preserve the *complete randomness* of their underlying Poisson processes. This yields a new Poisson process (due to theorem 1) and a related DP which depends on the source. In particular, we consider three such operations: *superposition*, *subsampling*, and *point transition*.

**Superposition of Poisson processes:** Combining a set of independent Poisson processes yields a Poisson process whose mean measure is the sum of mean measures of the individual ones.

**Theorem 2** (Superposition Theorem [9])**.** *Let $\Pi_1, \ldots, \Pi_m$ be independent Poisson processes on $\Omega$ with $\Pi_k \sim \mathrm{PoissonP}(\mu_k)$, then their union has*

$$\Pi_1 \cup \cdots \cup \Pi_m \sim \mathrm{PoissonP}(\mu_1 + \cdots + \mu_m). \tag{4}$$

Given a collection of independent Gamma processes $G_1, \ldots, G_m$, where for each $k = 1, \ldots, m$, $G_k \sim \Gamma\mathrm{P}(\mu_k)$ with underlying Poisson process $\Pi_k^* \sim \mathrm{PoissonP}(\mu_k \times \gamma)$. By theorem 2, we have

$$\bigcup_{k=1}^m \Pi_k^* \sim \mathrm{PoissonP}\left(\sum_{k=1}^m (\mu_k \times \gamma)\right) = \mathrm{PoissonP}\left(\left(\sum_{k=1}^m \mu_k\right) \times \gamma\right). \tag{5}$$

Due to the relationship between Gamma processes and their underlying Poisson processes, such a combination is equivalent to the direct superposition of the Gamma processes themselves, as

$$G' := G_1 + \cdots + G_m \sim \Gamma\mathrm{P}(\mu_1 + \cdots + \mu_m). \tag{6}$$

Let $D_k = G_k/G_k(\Omega)$, and $g_k = G_k(\Omega)$, then $D_k$ is independent of $g_k$, and thus

$$D' := G'/G'(\Omega) = (g_1 D_1 + \cdots + g_m D_m)/(g_1 + \cdots + g_m) = c_1 D_1 + \cdots + c_m D_m. \tag{7}$$

Here, $c_k = g_k / \sum_{l=1}^m g_l$, which has $(c_1, \ldots, c_m) \sim \mathrm{Dir}(\mu_1(\Omega), \ldots, \mu_m(\Omega))$. Consequently, *one can construct a Dirichlet process through a random convex combination of independent Dirichlet processes*. This result is summarized by the following theorem:

**Theorem 3.** *Let $D_1, \ldots, D_m$ be independent Dirichlet processes on $\Omega$ with $D_k \sim \mathrm{DP}(\mu_k)$, and $(c_1, \ldots, c_m) \sim \mathrm{Dir}(\mu_1(\Omega), \ldots, \mu_m(\Omega))$ be independent of $D_1, \ldots, D_m$, then*

$$D_1 \oplus \cdots \oplus D_m := c_1 D_1 + \cdots c_m D_m \sim \mathrm{DP}(\mu_1 + \cdots + \mu_m). \tag{8}$$

Here, we use the symbol $\oplus$ to indicate superposition via a random convex combination. Let $\alpha_k = \mu_k(\Omega)$ and $\alpha' = \sum_{k=1}^m \alpha_k$, then for each measurable subset $A$,

$$\mathbb{E}(D'(A)) = \sum_{k=1}^m \frac{\alpha_k}{\alpha'} \mathbb{E}(D_k(A)), \quad \text{and} \quad \mathrm{Cov}(D'(A), D_k(A)) = \frac{\alpha_k}{\alpha'} \mathrm{Var}(D_k(A)). \tag{9}$$

**Subsampling Poisson processes:** Random subsampling of a Poisson process via independent Bernoulli trials yields a new Poisson process.

**Theorem 4** (Subsampling Theorem). *Let $\Pi \sim \mathrm{PoissonP}(\mu)$ be a Poisson process on the space $\Omega$, and $q : \Omega \to [0, 1]$ be a measurable function. If we independently draw $z_\theta \in \{0, 1\}$ for each $\theta \in \Pi_0$ with $\mathbb{P}(z_\theta = 1) = q(\theta)$, and let $\Pi_k = \{\theta \in \Pi : z_\theta = k\}$ for $k = 0, 1$, then $\Pi_0$ and $\Pi_1$ are independent Poisson processes on $\Omega$, with $\Pi_0 \sim \mathrm{PoissonP}((1 - q)\mu)$ and $\Pi_1 \sim \mathrm{PoissonP}(q\mu)$*[1].

We emphasize that subsampling is via independent Bernoulli trials rather than choosing a fixed number of particles. We use $\mathcal{S}_q(\Pi) := \Pi_1$ to denote the result of subsampling, where $q$ is referred to as the *acceptance function*. Note that subsampling the underlying Poisson process of a Gamma process $G$ is equivalent to subsampling the terms of $G$. Let $G = \sum_{i=1}^\infty w_i \delta_{\theta_i}$, and for each $i$, we draw $z_i$ with $\mathbb{P}(z_i = 1) = q(\theta_i)$. Then, we have

$$G' = \mathcal{S}_q(G) := \sum_{i : z_i = 1} w_i \delta_{\theta_i} \sim \Gamma\mathrm{P}(q\mu). \tag{10}$$

Let $D$ be a Dirichlet process given by $D = G/G(\Omega)$, then we can construct a new Dirichlet process $D' = G'/G'(\Omega)$ by subsampling the terms of $D$ and renormalizing their coefficients. This is summarized by the following theorem.

**Theorem 5.** *Let $D \sim \mathrm{DP}(\mu)$ be represented by $D = \sum_{i=1}^n r_i \delta_{\theta_i}$ and $q : \Omega \to [0, 1]$ be a measurable function. For each $i$ we independently draw $z_i$ with $\mathbb{P}(z_i = 1) = q(\theta_i)$, then*

$$D' = \mathcal{S}_q(D) := \sum_{i : z_i = 1} r_i' \delta_{\theta_i} \sim \mathrm{DP}(q\mu), \tag{11}$$

*where $r_i' := r_i / \sum_{j : z_j = 1} r_j$ are the re-normalized coefficients for those $i$ with $z_i = 1$.*

Let $\alpha = \mu(\Omega)$ and $\alpha' = (q\mu)(\Omega)$, then for each measurable subset $A$,

$$\mathbb{E}(D'(A)) = \frac{(q\mu)(A)}{(q\mu)(\Omega)} = \frac{\int_A q \, d\mu}{\int_\Omega q \, d\mu}, \quad \text{and} \quad \mathrm{Cov}(D'(A), D(A)) = \frac{\alpha'}{\alpha} \mathrm{Var}(D'(A)). \tag{12}$$

**Point transition of Poisson processes:** The third operation moves each point independently following a probabilistic transition. Formally, a *probabilistic transition* is defined to be a function $T : \Omega \times \mathcal{F}_\Omega \to [0, 1]$ such that for each $\theta \in \mathcal{F}_\Omega$, $T(\theta, \cdot)$ is a probability measure on $\Omega$ that describes the distribution of where $\theta$ moves, and for each $A \in \mathcal{F}_\Omega$, $T(\cdot, A)$ is integrable. $T$ can be considered as a transformation of measures over $\Omega$, as

$$(T\mu)(A) := \int_\Omega T(\theta, A)\mu(d\theta). \tag{13}$$

**Theorem 6** (Transition Theorem). *Let* $\Pi \sim \text{PoissonP}(\mu)$ *and* $T$ *be a probabilistic transition, then*

$$T(\Pi) := \{T(\theta) : \theta \in \Pi\} \sim \text{PoissonP}(T\mu). \tag{14}$$

*With a slight abuse of notation, we use* $T(\theta)$ *to denote an independent sample from* $T(\theta, \cdot)$.

As a consequence, we can derive a Gamma process and thus a Dirichlet process by applying the probabilistic transition to the location of each term, leading to the following:

**Theorem 7.** *Let* $D = \sum_{i=1}^{\infty} r_i \delta_{\theta_i} \sim \text{DP}(\mu)$ *be a Dirichlet process on* $\Omega$, *then*

$$T(D) := \sum_{i=1}^{\infty} r_i \delta_{T(\theta_i)} \sim \text{DP}(T\mu). \tag{15}$$

Theorems 1 and 2 are immediate consequences of the results in [9]. We derive Theorems 3 to Theorem 7 independently as part of the proposed approach. Detailed explanation of relevant concepts and the proofs of Theorem 2 to Theorem 7 are provided in the supplement.

### 3.1 A Markov Chain of Dirichlet Processes

Integrating these three operations, we construct a Markov chain of DPs formulated as

$$D_t = T\left(\mathcal{S}_q(D_{t-1})\right) \oplus H_t, \qquad \text{with } H_t \sim \text{DP}(\nu). \tag{16}$$

The model can be explained as follows: given $D_{t-1}$, we choose a subset of terms by subsampling, then move their locations via a probabilistic transition $T$, and finally superimpose a new DP $H_t$ on the resultant process to form $D_t$. Hence, *creating new particles*, *removing existing particles*, and *varying particle locations* are all allowed, respectively, via *superposition*, *subsampling*, and *point transition*. Note that while they are based on the operations of the underlying Poisson processes, due to theorems 3, 5, and 7, we operate directly on the DPs, without the need of explicitly instantiating the associated Poisson processes or Gamma processes. Let $\mu_t$ be the base measure of $D_t$, then

$$\mu_t = T(q\mu_{t-1}) + \nu. \tag{17}$$

Particularly, if the acceptance probability $q$ is a constant, then $\alpha_t = q\alpha_{t-1} + \alpha_\nu$. Here, $\alpha_t = \mu_t(\Omega)$ and $\alpha_\nu = \nu(\Omega)$ are the concentration parameters. One may hold $\alpha_t$ fixed over time by choosing appropriate values for $q$ and $\alpha_\nu$. Furthermore, it can be shown that

$$\text{Cov}(D_{t+n}(A), D_t(A)) \leq q^n \text{Var}(D_t(A)). \tag{18}$$

The covariance with previous DPs decays exponentially when $q < 1$. This is often a desirable property in practice. Moreover, we note that $\nu$ and $q$ play different roles in controlling the process. Generally, $\nu$ determines how frequently new terms appear; while $q$ governs the life span of a term which has a geometric distribution with mean $(1-q)^{-1}$.

We aim to use the Markov chain of DPs as a prior of evolving mixture models. This provides a mechanism with which new component models can be brought in, existing components can be removed, and the model parameters can vary smoothly over time.

## 4 Comparison with Related Work

In his pioneering work [12], MacEachern proposed the "single-$p$ DDP model". It considers DDP as a collection of stochastic processes, but does not provide a natural mechanism to change the collection size over time. Müller et al [14] formulated each DP as a weighted mixture of a common DP and an independent DP. This formulation was extended by Dunson [6] in modeling latent trait distributions. Zhu et al [24] presented the Time-sensitive DP, in which the contribution of each DP decays exponentially. Teh et al [23] proposed the HDP where each child DP takes its parent DP as the base measure. Ren [18] combines the weighted mixture formulation with HDP to construct the dynamic HDP. In contrast to the model proposed here, a fundamental difference of these models is that the marginal distribution at each node is generally not a DP.

Caron et al [4] developed a generalized Polya Urn scheme while Ahmed and Xing [1] developed the recurrent Chinese Restaurant process (CRP). Both generalize the CRP to allow time-variation, while

retaining the property of being marginally DP. The motivation underlying these methods fundamentally differs from ours, leading to distinct differences in the sampling algorithm. In particular, [4] supports innovation and deletion of particles, but does not support variation of locations. Moreover, its deletion scheme is based on the distribution in history, but not on whether a component model fits the new observation. While [1] does support innovation and point transition, there is no explicit way to delete old particles. It can be considered a special case of the proposed framework in which subsampling operation is not incorporated. We note that [1] is motivated from an algorithmic rather than theoretical perspective.

Grifin and Steel [7] present the $\pi$DDP based on the stick breaking construction [19], reordering the stick breaking ratios for each time so as to obtain different distributions over the particles. This work is further extended [8] to a generic stick breaking processes. Chung et al [5] propose a local DP that generalizes $\pi$DDP. Rather than reordering the stick breaking ratios, they regroup them locally such that dependent DPs can be constructed over a general covariate space. Inference in these models requires sampling a series of auxiliary variables, considerably increasing computational costs. Moreover, the local DP relies on a truncated approximation to devise the sampling scheme.

Recently, Rao and Teh [16] proposed the spatially normalized Gamma process. They construct a universal Gamma process in an auxiliary space and obtain dependent DPs by normalizing it within overlapped local regions. The theoretical foundation differs in that it does not exploit the relationship between the Gamma and Poisson process which is at the heart of the proposed model. In [16], the dependency is established through region overlapping; while in our work, this is accomplished by explicitly transferring particles from one DP to another. In addition, this work does not support location variation, as it relies on a universal particle pool that is fixed over time.

## 5   The Sampling Algorithm

We develop a Gibbs sampling procedure based on the construction of DDPs introduced above. The key idea is to derive sampling steps by exploiting the fact that our construction maintains the property of being marginally DP via connections to the underlying Poisson processes. Furthermore, the derived procedure unifies distinct aspects (innovation, removal, and transition) of our model. Let $D \sim \mathrm{DP}(\mu)$ be a Dirichlet process on $\Omega$. Then given a set of samples $\Phi \sim D$, in which $\phi_i$ appears $c_i$ times, we have $D|\Phi \sim \mathrm{DP}(\mu + c_1\delta_{\phi_1} + \cdots + c_n\delta_{\phi_n})$. Let $D'$ be a Dirichlet process depending on $D$ as in Eq.(16), $\alpha_0 = (q\mu)(\Omega)$, and $q_i = q(\theta_i)$. Given $\Phi \sim D$, we have

$$D'|\Phi \sim \mathrm{DP}\left(\alpha_\nu p_\nu + \alpha_0 p_{q\mu} + \sum_{k=1}^{m} q_k c_k T(\phi_k, \cdot)\right). \tag{19}$$

**Sampling from $D'$.**   Let $\theta_1 \sim D'$. Marginalizing over $D'$, we get

$$\theta_1|\Phi \sim \frac{\alpha_\nu}{\alpha_1'} p_\nu + \frac{\alpha_0}{\alpha_1'} p_{q\mu} + \sum_{k=1}^{m} \frac{q_k c_k}{\alpha_1'} T(\phi_k, \cdot) \quad \text{with } \alpha_1' = \alpha_\nu + \alpha_0 + \sum_{k=1}^{m} q_k c_k. \tag{20}$$

Thus we sample $\theta_1$ from three types of sources: the innovation distribution $p_\nu$, the $q$-subsampled base distribution $p_{q\mu}$, and the transition distribution $T(\phi_k, \cdot)$. In doing so, we first sample a variable $u_1$ that indicates which source to sample from. Specifically, when $u_1 = -1$, $u_1 = 0$, or $u_1 = l > 0$, we respectively sample $\theta_1$ from $p_\nu$, $p_{q\mu}$, or $T(\phi_l, \cdot)$. The probabilities of these cases are $\alpha_\nu/\alpha_1'$, $\alpha_0/\alpha_1'$, and $q_i c_i/\alpha_1'$ respectively. After $u_1$ is obtained, we then draw $\theta_1$ from the indicated source. The next issue is *how to update the posterior given $\theta_1$ and $u_1$*. The answer depends on the value of $u_1$. When $u_1 = -1$ or $0$, $\theta_1$ is a new particle, and we have

$$D'|\theta_1, \{u_1 \le 0\} \sim \mathrm{DP}\left(\alpha_\nu p_\nu + \alpha_0 p_{q\mu} + \sum_{k=1}^{m} q_k c_k T(\phi_k, \cdot) + \delta_{\theta_1}\right). \tag{21}$$

If $u_1 = l > 0$, we know that the particle $\phi_l$ is retained in the subsampling process (i.e. the corresponding Bernoulli trial outputs 1), and the transited version $T(\phi_l)$ is determined to be $\theta_1$. Hence,

$$D'|\theta_1, \{u_1 = l > 0\} \sim \mathrm{DP}\left(\alpha_\nu p_\nu + \alpha_0 p_{q\mu} + \sum_{k \ne l} q_k c_k T(\theta_k, \cdot) + (c_l + 1)\delta_{\theta_1}\right). \tag{22}$$

With this posterior distribution, we can subsequently draw the second sample and so on. This process generalizes the Chinese restaurant process in several ways: (1) it allows either inheriting previous particles or drawing new ones; (2) it uses $q_k$ to control the chance that we sample a previous particle; (3) the transition $T$ allows smooth variation when we inherit a previous particle.

**Inference with Mixture Models.** We use the Markov chain of DPs as the prior of evolving mixture models. The generation process is formulated as

$$\theta_1, \ldots, \theta_n \sim D' \text{ i.i.d.,} \qquad \text{and} \qquad x_i \sim L(\theta_i), \ \ i = 1, \ldots, n. \tag{23}$$

Here, $L(\theta_i)$ is the observation model parameterized by $\theta_i$. According to the analysis above, we derive an algorithm to sample $\theta_1, \ldots, \theta_n$ conditioned on the observations $x_1, \ldots, x_n$ as follows.

**Initialization.** **(1)** Let $\tilde{m}$ denote *the number of particles*, which is initialized to be $m$ and will increase as we draw new particles from $p_\nu$ or $p_{q\mu}$. **(2)** Let $w_k$ denote *the prior weights* of different sampling sources which may also change during the sampling. Particularly, we set $w_k = q_k c_k$ for $k > 0$, $w_{-1} = \alpha_\nu$, and $w_0 = \alpha_0$. **(3)** Let $\psi_k$ denote *the particles*, whose value is decided when a new particle or the transited version of a previous one is sampled. **(4)** *The label $l_i$* indicates to which particle $\theta_i$ corresponds and *the counter $r_k$* records the number of times that $\psi_k$ has been sampled (set to 0 initially). **(5)** We compute *the expected likelihood*, as given by $F(k, i) := \mathbb{E}_{p_k}(f(x_i|\theta))$. Here, $f(x_i|\theta)$ is the likelihood of $x_j$ with respect to the parameter $\theta$, and $p_k$ is $p_\nu$, $p_{q\mu}$ or $T(\phi_k, \cdot)$ respectively when $k = -1$, $k = 0$ and $k \geq 1$.

**Sequential Sampling.** For each $i = 1, \ldots, n$, we first draw the indicator $u_i$ with probability $\mathbb{P}(u_i = k) \propto w_k F(k, i)$. Depending on the value of $u_i$, we sample $\theta_i$ from different sources. For brevity, let $p|x$ to denote the posterior distribution derived from the prior distribution $p$ conditioned on the observation $x$. **(1)** If $u_i = -1$ or 0, we draw $\theta_i$ from $p_\nu|x_i$ or $p_{qu}|x_i$, respectively, and then add it as a new particle. Concretely, we increase $\tilde{m}$ by 1, let $\psi_{\tilde{m}} = \theta_j$, $r_{\tilde{m}} = w_{\tilde{m}} = 1$, and set $l_i = \tilde{m}$. Moreover, we compute $F(m, i) = f(x_i|\psi_{\tilde{m}})$ for each $i$. **(2)** Suppose $u_i = k > 0$. If $r_k = 0$ then it is the first time we have drawn $u_i = k$. Since $\psi_k$ has not been determined, we sample $\theta_i \sim T(\phi_k, \cdot)|x_i$, then set $\psi_k = \theta_i$. If $r_k > 0$, the $k$-th particle has been sampled before. Thus, we can simply set $\theta_i = \psi_k$. In both cases, we set the label $l_i = k$, increase the weight $w_i$ and the counter $r_i$ by 1, and update $F(k, i)$ to $f(x_i|\psi_k)$ for each $i$.

Note that this procedure is inefficient in that it samples each particle $\phi_k$ merely based on the first observation with label $k$. Therefore, we use this procedure for bootstrapping, and then run a Gibbs sampling scheme that iterates between *parameter update* and *label update*.

**(Parameter update):** We resample each particle $\psi_k$ from its source distribution conditioned on all samples with label $k$. In particular, for $k \in [1, m]$ with $r_k > 0$, we draw $\psi_k \sim T(\phi_k, \cdot)|\{x_i : l_i = k\}$, and for $k \in [m + 1, \tilde{m}]$, we draw $\psi_k \sim p|\{x_i : l_i = k\}$, where $p = p_{qu}$ or $p_\nu$, depending which source $\psi_k$ was initially sampled from. After updating $\psi_k$, we need to update $F(k, i)$ accordingly.

**(Label update):** The label updating is similar to the bootstrapping procedure described above. The only difference is that when we update a label from $k$ to $k'$, we need to decrease the weight and counter for $k$. If $r_k$ decreases to zero, we remove $\psi_k$, and reset $w_k$ to $q_k c_k$ when $k \leq m$.

At the end of each phase $t$, we sample $\psi_k \sim T(\phi_k, \cdot)$ for each $k$ with $r_k = 0$. In addition, for each such particle, we update the acceptance probability as $q_k \leftarrow q_k \cdot q(\phi_k)$, which is the prior probability that the particle $\phi_k$ will survive in next phase. MATLAB code is available in the following website: http://code.google.com/p/ddpinfer/.

# 6 Experimental Results

Here we present experimental results on both synthetic and real data. In the synthetic case, we compare our method with dynamic FMM in modeling mixtures of Gaussians whose number and centers evolve over time. For real data, we test the approach in modeling the motion of people in crowded scenes and the trends of research topics reflected in index terms.

## 6.1 Simulations on Synthetic Data

The data for simulations were synthesized as follows. We initialized the model with two Gaussian components, and added new components following a temporal Poisson process (one per 20 phases

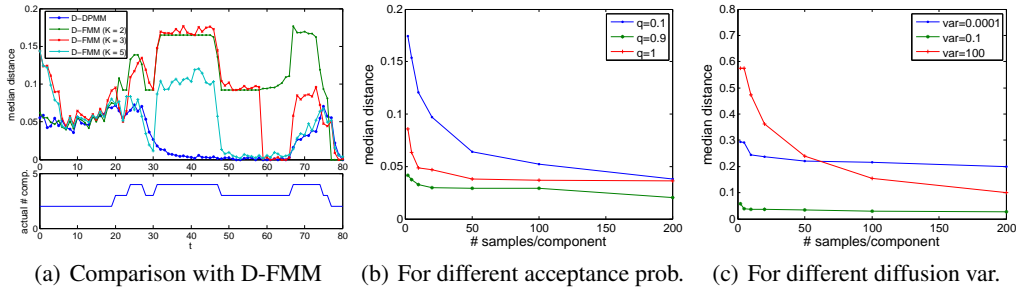

(a) Comparison with D-FMM      (b) For different acceptance prob.      (c) For different diffusion var.

Figure 1: The simulation results: (a) compares the performance between D-DPMM and D-FMM with differing numbers of components. The upper graph shows the median of distance between the resulting clusters and the ground truth at each phase. The lower graph shows the actual numbers of clusters. (b) shows the performance of D-DPMM with different values of acceptance probability, under different data sizes. (c) shows the performance of D-DPMM with different values of diffusion variance, under different data sizes.

on average). For each component, the life span has a geometric distribution with mean $40$, the mean evolves independently as a Brownian motion, and the variance is fixed to $1$. We performed the simulation for 80 phases, and at each phase, we drew 1000 samples for each active component. At each phase, we sample for 5000 iterations, discarding the first 2000 for burn-in, and collecting a sample every 100 iterations for performance evaluation. The particles of the last iteration at each phase were incorporated into the model as a prior for sampling in the next phase. We obtained the label for each observation by majority voting based on the collected samples, and evaluated the performance by measuring the dissimilarity between the resultant clusters and the ground truth using the *variation of information* [13] criterion. Under each parameter setting, we repeated the experiment 20 times, utilizing the median of the dissimilarities for comparison.

We compare our approach (D-DPMM) with dynamic finite mixtures (D-FMM), which assumes a fixed number of Gaussians whose centers vary as Brownian motion. From Figure 1(a), we observe that when the fixed number $K$ of components equals the actual number, they yield comparable performance; while when they are not equal, the errors of D-FMM substantially increase. Particularly, $K$ less than the actual number results in significant underfitting (e.g. D-FMM with $K = 2$ or $3$ at phases $30-50$ and $66-76$); when $K$ is greater than the actual number, samples from the same component are divided into multiple groups and assigned to different components (e.g. D-FMM with $K = 5$ at phases $1-10$ and $30-50$). In all cases, D-DPMM consistently outperforms D-FMM due to its ability to adjust the number of components to adapt to the change of observations.

We also studied how design parameters impact performance. In Figure 1(b), we see that an acceptance probability $q$ to $0.1$ creates new components rather than inheriting from previous phases, leading to poor performance when the number of samples is limited. If we set $q = 0.9$, the components in previous phases have a higher survival rate, resulting in more reliable estimation of the component parameters from multiple phases. Figure 1(c) shows the effect of the diffusion variance that controls the parameter variation. When it is small, the parameter in the next phase is tied tightly with the previous value; when it is large, the estimation basically relies on new observations. Both cases lead to performance degradation on small datasets, which indicates that it is important to maintain a balance between inheritance and innovation. Our framework provides the flexibility to attain such a balance. Cross-validation can be used to set these parameters automatically.

## 6.2 Real Data Applications

**Modeling People Flows.** It was observed [11] that the majority of people walking in crowded areas such as a rail station tend to follow motion flows. Typically, there are several flows at a time, and each flow may last for a period. In this experiment, we apply our approach to extract the flows. The test was conducted on video acquired in New York Grand Central Station, which comprises $90,000$ frames for one hour (25 fps). A low level tracker was used to obtain the tracks of people, which were then processed by a rule-based filter that discards obviously incorrect tracks. We adopt the flow model described in [11], which uses an affine field to capture the motion patterns of each flow. The observation for this model is in the form of location-velocity pairs. We divided the entire

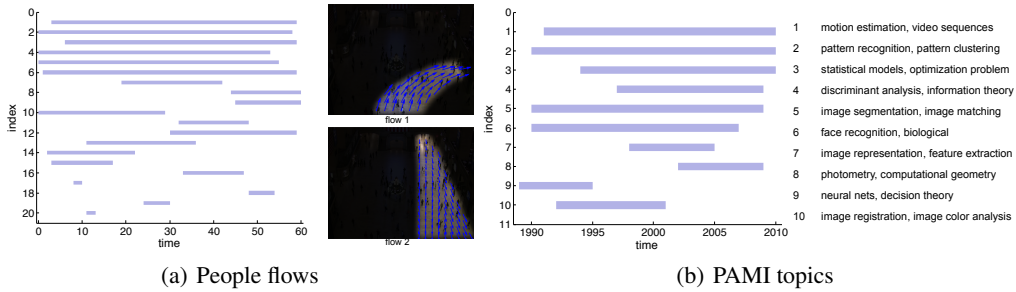

|  | (a) People flows | (b) PAMI topics |

Figure 2: The experiment results on real data. (a) left: the timelines of the top 20 flows; right: illustration of first two flows. (Illustrations of larger sizes are in the supplement.) (b) left: the timelines of the top 10 topics; right: the two leading keywords for these topics. (A list with more keywords is in the supplement.)

sequence into 60 phases (each for one minute), extract location-velocity pairs from all tracks, and randomly choose 3000 pairs for each phase for model inference. The algorithm infers 37 flows in total, while at each phase, the numbers of active flows range from 10 to 18. Figure 2(a) shows the timelines of the top 20 flows (in terms of the numbers of assigned observations). We compare the performance of our method with D-FMM by measuring the average likelihood on a disjoint dataset. The value for our method is $-3.34$, while those for D-FMM are $-6.71$, $-5.09$, $-3.99$, $-3.49$, and $-3.34$, when $K$ are respectively set to $10, 20, 30, 40$, and $50$. Consequently, with a much smaller number of components (12 active components on average), our method attains a similar modeling accuracy as a D-FMM with 50 components.

**Modeling Paper Topics.** Next we analyze the evolution of paper topics for IEEE Trans. on PAMI. By parsing the webpage of IEEE Xplore, we collected the index terms for 3014 papers published in PAMI from Jan, 1990 to May, 2010. We first compute the similarity between each pair of papers in terms of relative fraction of overlapped index terms. We derive a 12-dimensional feature vector using spectral embedding [2] over the similarity matrix for each paper. We run our algorithm on these features with each phase corresponding to a year. Each cluster of papers is deemed a *topic*. We compute the histogram of index terms and sorted them in decreasing order of frequency for each topic. Figure 2(b) shows the timelines of top 10 topics, and together with the top two index terms for each of them. Not surprisingly, we see that topics such as "neural networks" arise early and then diminish while "image segmentation" and "motion estimation" persist.

# 7  Conclusion and Future Directions

We developed a principled framework for constructing dependent Dirichlet processes. In contrast to most DP-based approaches, our construction is motivated by the intrinsic relation between Dirichlet processes and compound Poisson processes. In particular, we discussed three operations: super-position, subsampling, and point transition, which produce DPs depending on others. We further combined these operations to derive a Markov chain of DPs, leading to a prior of mixture models that allows creation, removal, and location variation of component models under a unified formulation. We also presented a Gibbs sampling algorithm for inferring the models. The simulations on synthetic data and the experiments on modeling people flows and paper topics clearly demonstrate that the proposed method is effective in estimating mixture models that evolve over time.

This framework can be further extended along different directions. The fact that each completely random point process is a Poisson process suggests that any operation that preserves the complete randomness can be applied to obtain dependent Poisson processes, and thus dependent DPs. Such operations are definitely not restricted to the three ones discussed in this paper. For example, random merging and random splitting of particles also possess this property, which would lead to an extended framework that allows merging and splitting of component models. Furthermore, while we focused on Markov chain in this paper, the framework can be straightforwardly generalized to any acyclic network of DPs. It is also interesting to study how it can be generalized to the case with undirected network or even continuous covariate space. We believe that as a starting point, this paper would stimulate further efforts to exploit the relation between Poisson processes and Dirichlet processes.

## Footnotes

[1]$q\mu$ is a measure on $\Omega$ given by $(q\mu)(A) = \int_A q \, d\mu$, or equivalently $(q\mu)(d\theta) = q(\theta)\mu(d\theta)$.

# References

[1] A. Ahmed and E. Xing. Dynamic Non-Parametric Mixture Models and The Recurrent Chinese Restaurant Process : with Applications to Evolutionary Clustering. In *Proc. of SDM'08*, 2008.

[2] F. R. Bach and M. I. Jordan. Learning spectral clustering. In *Proc. of NIPS'03*, 2003.

[3] J. Boyd-Graber and D. M. Blei. Syntactic Topic Models. In *Proc. of NIPS'08*, 2008.

[4] F. Caron, M. Davy, and A. Doucet. Generalized Polya Urn for Time-varying Dirichlet Process Mixtures. In *Proc. of UAI'07*, number 6, 2007.

[5] Y. Chung and D. B. Dunson. The local Dirichlet Process. *Annals of the Inst. of Stat. Math.*, (October 2007), January 2009.

[6] D. B. Dunson. Bayesian Dynamic Modeling of Latent Trait Distributions. *Biostatistics*, 7(4), October 2006.

[7] J. E. Griffin and M. F. J. Steel. Order-Based Dependent Dirichlet Processes. *Journal of the American Statistical Association*, 101(473):179–194, March 2006.

[8] J. E. Griffin and M. F. J. Steel. Time-Dependent Stick-Breaking Processes. Technical report, 2009.

[9] J. F. C. Kingman. *Poisson Processes*. Oxford University Press, 1993.

[10] J. J. Kivinen, E. B. Sudderth, and M. I. Jordan. Learning Multiscale Representations of Natural Scenes Using Dirichlet Processes. In *Proc. of ICCV'07*, 2007.

[11] D. Lin, E. Grimson, and J. Fisher. Learning Visual Flows: A Lie Algebraic Approach. In *Proc. of CVPR'09*, 2009.

[12] S. N. MacEachern. Dependent Nonparametric Processes. In *Proceedings of the Section on Bayesian Statistical Science*, 1999.

[13] M. Meila. Comparing clusterings - An Axiomatic View. In *Proc. of ICML'05*, 2005.

[14] P. Muller, F. Quintana, and G. Rosner. A Method for Combining Inference across Related Nonparametric Bayesian Models. *J. R. Statist. Soc. B*, 66(3):735–749, August 2004.

[15] R. M. Neal. Markov Chain Sampling Methods for Dirichlet Process Mixture Models. *Journal of computational and graphical statistics*, 9(2):249–265, 2000.

[16] V. Rao and Y. W. Teh. Spatial Normalized Gamma Processes. In *Proc. of NIPS'09*, 2009.

[17] C. E. Rasmussen. The Infinite Gaussian Mixture Model. In *Proc. of NIPS'00*, 2000.

[18] L. Ren, D. B. Dunson, and L. Carin. The Dynamic Hierarchical Dirichlet Process. In *Proc. of ICML'08*, New York, New York, USA, 2008. ACM Press.

[19] J. Sethuraman. A Constructive Definition of Dirichlet Priors. *Statistica Sinica*, 4(2):639–650, 1994.

[20] K.-a. Sohn and E. Xing. Hidden Markov Dirichlet process: modeling genetic recombination in open ancestral space. In *Proc. of NIPS'07*, 2007.

[21] N. Srebro and S. Roweis. Time-Varying Topic Models using Dependent Dirichlet Processes, 2005.

[22] Y. W. Teh. Dirichlet Process, 2007.

[23] Y. W. Teh, M. I. Jordan, M. J. Beal, and D. M. Blei. Hierarchical Dirichlet Processes. *Journal of the American Statistical Association*, 101(476):1566–1581, 2006.

[24] X. Zhu and J. Lafferty. Time-Sensitive Dirichlet Process Mixture Models, 2005.

